# Nyström Method vs Random Fourier Features: A Theoretical and Empirical Comparison

**Tianbao Yang**[†], **Yu-Feng Li**[‡], **Mehrdad Mahdavi**[♮], **Rong Jin**[♮], **Zhi-Hua Zhou**[‡]
[†]Machine Learning Lab, GE Global Research, San Ramon, CA 94583
[♮]Michigan State University, East Lansing, MI 48824
[‡]National Key Laboratory for Novel Software Technology, Nanjing University, 210023, China
tyang@ge.com,mahdavim,rongjin@msu.edu,liyf,zhouzh@lamda.nju.edu.cn

## Abstract

Both random Fourier features and the Nyström method have been successfully applied to efficient kernel learning. In this work, we investigate the fundamental difference between these two approaches, and how the difference could affect their generalization performances. Unlike approaches based on random Fourier features where the basis functions (i.e., cosine and sine functions) are sampled from a distribution *independent* from the training data, basis functions used by the Nyström method are randomly sampled from the training examples and are therefore *data dependent*. By exploring this difference, we show that when there is a large gap in the eigen-spectrum of the kernel matrix, approaches based on the Nyström method can yield impressively better generalization error bound than random Fourier features based approach. We empirically verify our theoretical findings on a wide range of large data sets.

## 1 Introduction

Kernel methods [16], such as support vector machines, are among the most effective learning methods. These methods project data points into a high-dimensional or even infinite-dimensional feature space and find the optimal hyperplane in that feature space with strong generalization performance. One limitation of kernel methods is their high computational cost, which is at least quadratic in the number of training examples, due to the calculation of kernel matrix. Although low rank decomposition approaches (e.g., incomplete Cholesky decomposition [3]) have been used to alleviate the computational challenge of kernel methods, they still require computing the kernel matrix. Other approaches such as online learning [9] and budget learning [7] have also been developed for large-scale kernel learning, but they tend to yield performance worse performance than batch learning.

To avoid computing kernel matrix, one common approach is to approximate a kernel learning problem with a linear prediction problem. It is often achieved by generating a vector representation of data that approximates the kernel similarity between any two data points. The most well known approaches in this category are *random Fourier features* [13, 14] and *the Nyström method* [20, 8]. Although both approaches have been found effective, it is not clear what are their essential difference, and which method is preferable under which situations. The objective of this work is to understand the difference between these two approaches, both theoretically and empirically

The theoretical foundation for random Fourier transform is that a shift-invariant kernel is the Fourier transform of a non-negative measure [15]. Using this property, in [13], the authors proposed to represent each data point by random Fourier features. Analysis in [14] shows that, the generalization error bound for kernel learning based on random Fourier features is given by $O(N^{-1/2} + m^{-1/2})$, where $N$ is the number of training examples and $m$ is the number of sampled Fourier components.

An alternative approach for large-scale kernel classification is the Nyström method [20, 8] that approximates the kernel matrix by a low rank matrix. It randomly samples a subset of training examples and computes a kernel matrix $\widehat{K}$ for the random samples. It then represents each data point by a vector based on its kernel similarity to the random samples and the sampled kernel matrix $\widehat{K}$. Most analysis of the Nyström method follows [8] and bounds the error in approximating the kernel matrix. According to [8], the approximation error of the Nyström method, measured in spectral norm [1], is $O(m^{-1/2})$, where $m$ is the number of sampled training examples. Using the arguments in [6], we expected an additional error of $O(m^{-1/2})$ in the generalization performance caused by the approximation of the Nyström method, similar to random Fourier features.

**Contributions** In this work, we first establish a unified framework for both methods from the viewpoint of functional approximation. This is important because random Fourier features and the Nyström method address large-scale kernel learning very differently: random Fourier features aim to approximate the kernel function directly while the Nyström method is designed to approximate the kernel matrix. The unified framework allows us to see a fundamental difference between the two methods: the basis functions used by random Fourier features are randomly sampled from a distribution *independent* from the training data, leading to a data independent vector representation; in contrast, the Nyström method randomly selects a subset of training examples to form its basis functions, leading to a data dependent vector representation. By exploring this difference, we show that the additional error caused by the Nyström method in the generalization performance can be improved to $O(1/m)$ when there is a large gap in the eigen-spectrum of the kernel matrix. Empirical studies on a synthetic data set and a broad range of real data sets verify our analysis.

## 2 A Unified Framework for Approximate Large-Scale Kernel Learning

Let $\mathcal{D} = \{(\mathbf{x}_1, y_1), \ldots, (\mathbf{x}_N, y_N)\}$ be a collection of $N$ training examples, where $\mathbf{x}_i \in \mathcal{X} \subseteq \mathbb{R}^d$, $y_i \in \mathcal{Y}$. Let $\kappa(\cdot, \cdot)$ be a kernel function, $\mathcal{H}_\kappa$ denote the endowed Reproducing Kernel Hilbert Space, and $K = [\kappa(\mathbf{x}_i, \mathbf{x}_j)]_{N \times N}$ be the kernel matrix for the samples in $\mathcal{D}$. Without loss of generality, we assume $\kappa(\mathbf{x}, \mathbf{x}) \leq 1, \forall \mathbf{x} \in \mathcal{X}$. Let $(\lambda_i, \mathbf{v}_i), i = 1, \ldots, N$ be the eigenvalues and eigenvectors of $K$ ranked in the descending order of eigenvalues. Let $V = [V_{ij}]_{N \times N} = (\mathbf{v}_1, \ldots, \mathbf{v}_N)$ denote the eigenvector matrix. For the Nyström method, let $\widehat{\mathcal{D}} = \{\widehat{\mathbf{x}}_1, \ldots, \widehat{\mathbf{x}}_m\}$ denote the randomly sampled examples, $\widehat{K} = [\kappa(\widehat{\mathbf{x}}_i, \widehat{\mathbf{x}}_j)]_{m \times m}$ denote the corresponding kernel matrix. Similarly, let $\{(\widehat{\lambda}_i, \widehat{\mathbf{v}}_i), i \in [m]\}$ denote the eigenpairs of $\widehat{K}$ ranked in the descending order of eigenvalues, and $\widehat{V} = [\widehat{V}_{ij}]_{m \times m} = (\widehat{\mathbf{v}}_1, \ldots, \widehat{\mathbf{v}}_m)$. We introduce two linear operators induced by examples in $\mathcal{D}$ and $\widehat{\mathcal{D}}$, i.e.,

$$L_N[f] = \frac{1}{N} \sum_{i=1}^{N} \kappa(\mathbf{x}_i, \cdot) f(\mathbf{x}_i), \quad L_m[f] = \frac{1}{m} \sum_{i=1}^{m} \kappa(\widehat{\mathbf{x}}_i, \cdot) f(\widehat{\mathbf{x}}_i). \tag{1}$$

It can be shown that both $L_N$ and $L_m$ are self-adjoint operators. According to [18], the eigenvalues of $L_N$ and $L_m$ are $\lambda_i/N, i \in [N]$ and $\widehat{\lambda}_i/m, i \in [m]$, respectively, and their corresponding normalized eigenfunctions $\varphi_j, j \in [N]$ and $\widehat{\varphi}_j, j \in [m]$ are given by

$$\varphi_j(\cdot) = \frac{1}{\sqrt{\lambda_j}} \sum_{i=1}^{N} V_{i,j} \kappa(\mathbf{x}_i, \cdot), \ j \in [N], \quad \widehat{\varphi}_j(\cdot) = \frac{1}{\sqrt{\widehat{\lambda}_j}} \sum_{i=1}^{m} \widehat{V}_{i,j} \kappa(\widehat{\mathbf{x}}_i, \cdot), j \in [m]. \tag{2}$$

To make our discussion concrete, we focus on the RBF kernel [2], i.e., $\kappa(\mathbf{x}, \bar{\mathbf{x}}) = \exp(-\|\mathbf{x} - \bar{\mathbf{x}}\|_2^2/[2\sigma^2])$, whose inverse Fourier transform is given by a Gaussian distribution $p(\mathbf{u}) = \mathcal{N}(0, \sigma^{-2}I)$ [15]. Our goal is to efficiently learn a kernel prediction function by solving the following optimization problem:

$$\min_{f \in \mathcal{H}_{\mathcal{D}}} \frac{\lambda}{2} \|f\|_{\mathcal{H}_\kappa}^2 + \frac{1}{N} \sum_{i=1}^{N} \ell(f(\mathbf{x}_i), y_i), \tag{3}$$

where $\mathcal{H}_\mathcal{D} = \mathrm{span}(\kappa(\mathbf{x}_1, \cdot), \ldots, \kappa(\mathbf{x}_N, \cdot))$ is a span over all the training examples [3], and $\ell(z, y)$ is a convex loss function with respect to $z$. To facilitate our analysis, we assume $\max_{y \in \mathcal{Y}} \ell(0, y) \le 1$ and $\ell(z, y)$ has a bounded gradient $|\nabla_z \ell(z, y)| \le C$. The high computational cost of kernel learning arises from the fact that we have to search for an optimal classifier $f(\cdot)$ in a *large* space $\mathcal{H}_\mathcal{D}$.

Given this observation, to alleviate the computational cost of kernel classification, we can reduce space $\mathcal{H}_\mathcal{D}$ to a smaller space $\mathcal{H}_a$, and only search for the solution $f(\cdot) \in \mathcal{H}_a$. The main challenge is how to construct such a space $\mathcal{H}_a$. On the one hand, $\mathcal{H}_a$ should be small enough to make it possible to perform efficient computation; on the other hand, $\mathcal{H}_a$ should be rich enough to provide good approximation for most bounded functions in $\mathcal{H}_\mathcal{D}$. Below we show that the difference between random Fourier features and the Nyström method lies in the construction of the approximate space $\mathcal{H}_a$. For each method, we begin with a description of a vector representation of data, and then connect the vector representation to the approximate large kernel machine by functional approximation.

**Random Fourier Features** The random Fourier features are constructed by first sampling Fourier components $\mathbf{u}_1, \ldots, \mathbf{u}_m$ from $p(\mathbf{u})$, projecting each example $\mathbf{x}$ to $\mathbf{u}_1, \ldots, \mathbf{u}_m$ separately, and then passing them through sine and cosine functions, i.e., $\mathbf{z}_f(\mathbf{x}) = (sin(\mathbf{u}_1^\top \mathbf{x}), cos(\mathbf{u}_1^\top \mathbf{x}), \ldots, sin(\mathbf{u}_m^\top \mathbf{x}), cos(\mathbf{u}_m^\top \mathbf{x}))$. Given the random Fourier features, we then learn a linear machine $f(\mathbf{x}) = \mathbf{w}^\top \mathbf{z}_f(\mathbf{x})$ by solving the following optimization problem:

$$\min_{\mathbf{w} \in \mathbb{R}^{2m}} \frac{\lambda}{2} \|\mathbf{w}\|_2^2 + \frac{1}{N} \sum_{i=1}^N \ell(\mathbf{w}^\top \mathbf{z}_f(\mathbf{x}_i), y_i). \tag{4}$$

To connect the linear machine (4) to the kernel machine in (3) by a functional approximation, we can construct a functional space $\mathcal{H}_a^f = \mathrm{span}(s_1(\cdot), c_1(\cdot), \ldots, s_m(\cdot), c_m(\cdot))$, where $s_k(\mathbf{x}) = \sin(\mathbf{u}_k^\top \mathbf{x})$ and $c_k(\mathbf{x}) = \cos(\mathbf{u}_k^\top \mathbf{x})$. If we approximate $\mathcal{H}_\mathcal{D}$ in (3) by $\mathcal{H}_a^f$, we have

$$\min_{f \in \mathcal{H}_a^f} \frac{\lambda}{2} \|f\|_{\mathcal{H}_\kappa}^2 + \frac{1}{N} \sum_{i=1}^N \ell(f(\mathbf{x}_i), y_i). \tag{5}$$

The following proposition connects the approximate kernel machine in (5) to the linear machine in (4). Proofs can be found in supplementary file.

**Proposition 1** *The approximate kernel machine in (5) is equivalent to the following linear machine*

$$\min_{\mathbf{w} \in \mathbb{R}^{2m}} \frac{\lambda}{2} \mathbf{w}^\top (\mathbf{w} \circ \gamma) + \frac{1}{N} \sum_{i=1}^N \ell(\mathbf{w}^\top \mathbf{z}_f(\mathbf{x}_i), y_i), \tag{6}$$

*where* $\gamma = (\gamma_1^s, \gamma_1^c, \cdots, \gamma_m^s, \gamma_m^c)^\top$ *and* $\gamma_i^{s/c} = \exp(\sigma^2 \|\mathbf{u}_i\|_2^2 / 2)$.

Comparing (6) to the linear machine based on random Fourier features in (4), we can see that other than the weights $\{\gamma_i^{s/c}\}_{i=1}^m$, random Fourier features can be viewed as to approximate (3) by restricting the solution $f(\cdot)$ to $\mathcal{H}_a^f$.

**The Nyström Method** The Nyström method approximates the full kernel matrix $K$ by first sampling $m$ examples, denoted by $\widehat{\mathbf{x}}_1, \cdots, \widehat{\mathbf{x}}_m$, and then constructing a low rank matrix by $\widehat{K}_r = K_b \widehat{K}^\dagger K_b^\top$, where $K_b = [\kappa(\mathbf{x}_i, \widehat{\mathbf{x}}_j)]_{N \times m}$, $\widehat{K} = [\kappa(\widehat{\mathbf{x}}_i, \widehat{\mathbf{x}}_j)]_{m \times m}$, $\widehat{K}^\dagger$ is the pseudo inverse of $\widehat{K}$, and $r$ denotes the rank of $\widehat{K}$. In order to train a linear machine, we can derive a vector representation of data by $\mathbf{z}_n(\mathbf{x}) = \widehat{D}_r^{-1/2} \widehat{V}_r^\top (\kappa(\mathbf{x}, \widehat{\mathbf{x}}_1), \ldots, \kappa(\mathbf{x}, \widehat{\mathbf{x}}_m))^\top$, where $\widehat{D}_r = diag(\widehat{\lambda}_1, \ldots, \widehat{\lambda}_r)$ and $\widehat{V}_r = (\widehat{\mathbf{v}}_1, \ldots, \widehat{\mathbf{v}}_r)$. It is straightforward to verify that $\mathbf{z}_n(\mathbf{x}_i)^\top \mathbf{z}_n(\mathbf{x}_j) = [\widehat{K}_r]_{ij}$. Given the vector representation $\mathbf{z}_n(\mathbf{x})$, we then learn a linear machine $f(\mathbf{x}) = \mathbf{w}^\top \mathbf{z}_n(\mathbf{x})$ by solving the following optimization problem:

$$\min_{\mathbf{w} \in \mathbb{R}^r} \frac{\lambda}{2} \|\mathbf{w}\|_2^2 + \frac{1}{N} \sum_{i=1}^N \ell(\mathbf{w}^\top \mathbf{z}_n(\mathbf{x}_i), y_i). \tag{7}$$

In order to see how the Nyström method can be cast into the unified framework of approximating the large scale kernel machine by functional approximation, we construct the following functional space $\mathcal{H}_a^n = \mathrm{span}(\widehat{\varphi}_1, \ldots, \widehat{\varphi}_r)$, where $\widehat{\varphi}_1, \ldots, \widehat{\varphi}_r$ are the first $r$ normalized eigenfunctions of the operator $L_m$. The following proposition shows that the linear machine in (7) using the vector representation of the Nyström method is equivalent to the approximate kernel machine in (3) by restricting the solution $f(\cdot)$ to an approximate functional space $\mathcal{H}_a^n$.

**Proposition 2** *The linear machine in (7) is equivalent to the following approximate kernel machine*

$$\min_{f \in \mathcal{H}_a^n} \frac{\lambda}{2} \|f\|_{\mathcal{H}_\kappa}^2 + \frac{1}{N} \sum_{i=1}^{N} \ell(f(\mathbf{x}_i), y_i), \tag{8}$$

Although both random Fourier features and the Nyström method can be viewed as variants of the unified framework, they differ significantly in the construction of the approximate functional space $\mathcal{H}_a$. In particular, the basis functions used by random Fourier features are sampled from a Gaussian distribution that is independent from the training examples. In contrast, the basis functions used by the Nyström method are sampled from the training examples and are therefore data dependent.

This difference, although subtle, can have significant impact on the classification performance. In the case of large eigengap, i.e., the first few eigenvalues of the full kernel matrix are much larger than the remaining eigenvalues, the classification performance is mostly determined by the top eigenvectors. Since the Nyström method uses a data dependent sampling method, it is able to discover the subspace spanned by the top eigenvectors using a small number of samples. In contrast, since random Fourier features are drawn from a distribution independent from training data, it may require a large number of samples before it can discover this subspace. As a result, we expect a significantly lower generalization error for the Nyström method.

To illustrate this point, we generate a synthetic data set consisted of two balanced classes with a total of $N = 10,000$ data points generated from uniform distributions in two balls of radius $0.5$ centered at $(-0.5, 0.5)$ and $(0.5, 0.5)$, respectively. The $\sigma$ value in the RBF kernel is chosen by cross-validation and is set to 6 for the synthetic data. To avoid a trivial task, 100 redundant features, each drawn from a uniform distribution on the unit interval, are added to each example. The data points in the first two dimensions are plotted in Figure 1(a) [4], and the eigenvalue distribution is shown in Figure 1(b). According to the results shown in Figure 1(c), it is clear that the Nyström method performs significantly better than random Fourier features. By using only 100 samples, the Nyström method is able to make perfect prediction, while the decision made by random Fourier features based method is close to random guess. To evaluate the approximation error of the functional space, we plot in Figure 1(e) and 1(f), respectively, the first two eigenvectors of the approximate kernel matrix computed by the Nyström method and random Fourier features using 100 samples. Compared to the eigenvectors computed from the full kernel matrix (Figure 1(d)), we can see that the Nyström method achieves a significantly better approximation of the first two eigenvectors than random Fourier features.

Finally, we note that although the concept of eigengap has been exploited in many studies of kernel learning [2, 12, 1, 17], to the best of our knowledge, this is the first time it has been incorporated in the analysis for approximate large-scale kernel learning.

## 3 Main Theoretical Result

Let $f_m^*$ be the optimal solution to the approximate kernel learning problem in (8), and let $f_N^*$ be the solution to the full version of kernel learning in (3). Let $f^*$ be the optimal solution to

$$\min_{f \in \mathcal{H}_\kappa} \left( F(f) = \frac{\lambda}{2} \|f\|_{\mathcal{H}_\kappa}^2 + \mathrm{E}\left[\ell(f(\mathbf{x}), y)\right] \right),$$

where $\mathrm{E}[\cdot]$ takes expectation over the joint distribution $P(\mathbf{x}, y)$. Following [10], we define the excess risk of any classifier $f \in \mathcal{H}_\kappa$ as

$$\Lambda(f) = F(f) - F(f^*). \tag{9}$$

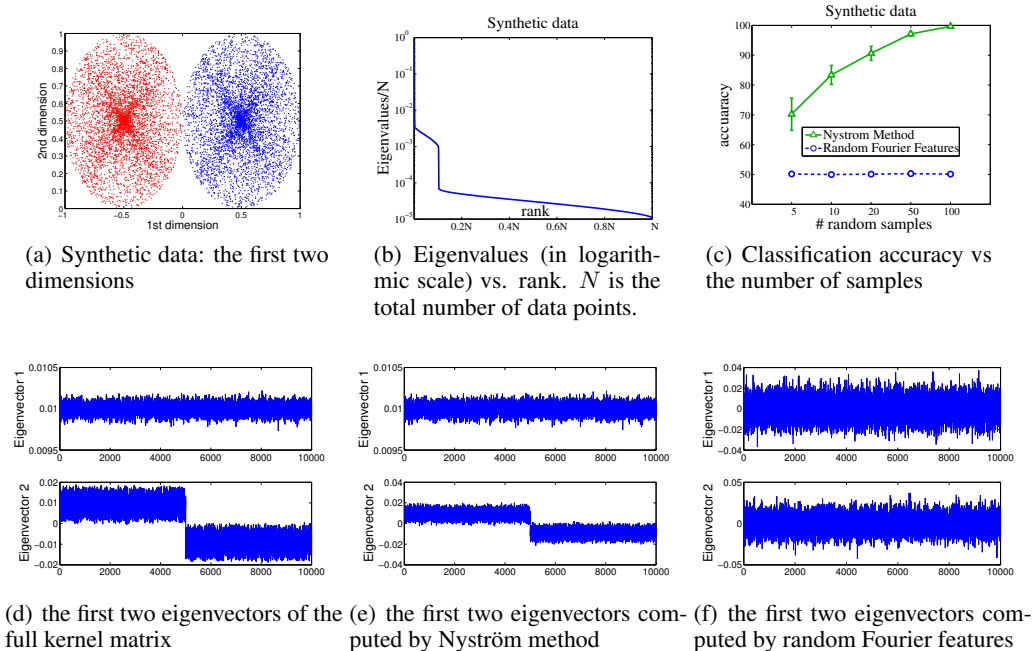

(a) Synthetic data: the first two dimensions

(b) Eigenvalues (in logarithmic scale) vs. rank. $N$ is the total number of data points.

(c) Classification accuracy vs the number of samples

(d) the first two eigenvectors of the full kernel matrix

(e) the first two eigenvectors computed by Nyström method

(f) the first two eigenvectors computed by random Fourier features

Figure 1: An Illustration Example

Unlike [6], in this work, we aim to bound the generalization performance of $f_m^*$ by the generalization performance of $f_N^*$, which better reflects the impact of approximating $\mathcal{H}_{\mathcal{D}}$ by $\mathcal{H}_a^n$.

In order to obtain a tight bound, we exploit the local Rademacher complexity [10]. Define $\psi(\delta) = \left( \frac{2}{N} \sum_{i=1}^N \min(\delta^2, \lambda_i) \right)^{1/2}$. Let $\widetilde{\varepsilon}$ as the solution to $\widetilde{\varepsilon}^2 = \psi(\widetilde{\varepsilon})$ where the existence and uniqueness of $\widetilde{\varepsilon}$ are determined by the sub-root property of $\psi(\delta)$ [4], and $\epsilon = \max\left( \widetilde{\varepsilon}, \sqrt{\frac{6 \ln N}{N}} \right)$. According to [10], we have $\epsilon^2 = O(N^{-1/2})$, and when the eigenvalues of kernel function follow a $p$-power law, it is improved to $\epsilon^2 = O(N^{-p/(p+1)})$. The following theorem bounds $\Lambda(f_m^*)$ by $\Lambda(f_N^*)$. Section 4 will be devoted to the proof of this theorem.

**Theorem 1** *For* $16\epsilon^2 e^{-2N} \leq \lambda \leq 1$, $\lambda_{r+1} = O(N/m)$ *and*

$$(\lambda_r - \lambda_{r+1})/N = \Omega(1) \geq 3 \left( \frac{2 \ln(2N^3)}{m} + \sqrt{\frac{2 \ln(2N^3)}{m}} \right),$$

*with a probability* $1 - 3N^{-3}$*, we have*

$$\Lambda(f_m^*) \leq 3\Lambda(f_N^*) + \frac{1}{\lambda} \widetilde{O}\left( \epsilon^2 + \frac{1}{m} \right),$$

*where* $\widetilde{O}(\cdot)$ *suppresses the polynomial term of* $\ln N$*.*

Theorem 1 shows that the additional error caused by the approximation of the Nyström method is improved to $O(1/m)$ when there is a large gap between $\lambda_r$ and $\lambda_{r+1}$. Note that the improvement from $O(1/\sqrt{m})$ to $O(1/m)$ is very significant from the theoretical viewpoint, because it is well known that the generalization error for kernel learning is $O(N^{-1/2})$ [4][5]. As a result, to achieve a similar performance as the standard kernel learning, the number of required samples has to be

$O(N)$ if the additional error caused by the kernel approximation is bounded by $O(1/\sqrt{m})$, leading to a high computational cost. On the other hand, with $O(1/m)$ bound for the additional error caused by the kernel approximation, the number of required samples is reduced to $\sqrt{N}$, making it more practical for large-scale kernel learning.

We also note that the improvement made for the Nyström method relies on the property that $\mathcal{H}_a^n \subset \mathcal{H}_\mathcal{D}$ and therefore requires data dependent basis functions. As a result, it does not carry over to random Fourier features.

## 4 Analysis

In this section, we present the analysis that leads to Theorem 1. Most of the proofs can be found in the supplementary materials. We first present a theorem to show that the excessive risk bound of $f_m^*$ is related to the matrix approximation error $\|K - \widehat{K}_r\|_2$.

**Theorem 2** *For $16\epsilon^2 e^{-2N} \leq \lambda \leq 1$, with a probability $1 - 2N^{-3}$, we have*

$$\Lambda(f_m^*) \leq 3\Lambda(f_N^*) + C_2 \left( \frac{\epsilon^2}{\lambda} + \frac{\|K - \widehat{K}_r\|_2}{N\lambda} + e^{-N} \right),$$

*where $C_2$ is a numerical constant.*

In the sequel, we let $K_r$ be the best rank-$r$ approximation matrix for $K$. By the triangle inequality, $\|K - \widehat{K}_r\|_2 \leq \|K - K_r\|_2 + \|K_r - \widehat{K}_r\|_2 \leq \lambda_{r+1} + \|K_r - \widehat{K}_r\|_2$, we thus proceed to bound $\|K_r - \widehat{K}_r\|_2$. Using the eigenfunctions of $L_m$ and $L_N$, we define two linear operators $H_r$ and $\widehat{H}_r$ as

$$H_r[f](\cdot) = \sum_{i=1}^r \varphi_i(\cdot)\langle\varphi_i, f\rangle_{\mathcal{H}_\kappa}, \quad \widehat{H}_r[f](\cdot) = \sum_{i=1}^r \widehat{\varphi}_i(\cdot)\langle\widehat{\varphi}_i, f\rangle_{\mathcal{H}_\kappa}, \tag{10}$$

where $f \in \mathcal{H}_\kappa$. The following theorem shows that $\|K_r - \widehat{K}_r\|_2$ is related to the linear operator $\Delta H = H_r - \widehat{H}_r$.

**Theorem 3** *For $\widehat{\lambda}_r > 0$ and $\lambda_r > 0$, we have*

$$\|\widehat{K}_r - K_r\|_2 \leq N\|L_N^{1/2}\Delta H L_N^{1/2}\|_2,$$

*where $\|L\|_2$ stands for the spectral norm of a linear operator $L$.*

Given the result in Theorem 3, we move to bound the spectral norm of $L_N^{1/2}\Delta H L_N^{1/2}$. To this end, we assume a sufficiently large eigengap $\Delta = (\lambda_r - \lambda_{r+1})/N$. The theorem below bounds $\|L_N^{1/2}\Delta H L_N^{1/2}\|_2$ using matrix perturbation theory [19].

**Theorem 4** *For $\Delta = (\lambda_r - \lambda_{r+1})/N > 3\|L_N - L_m\|_{HS}$, we have*

$$\|L_N^{1/2}\Delta H L_N^{1/2}\|_2 \leq \eta \frac{4\|L_N - L_m\|_{HS}}{\Delta - \|L_N - L_m\|_{HS}},$$

*where $\eta = \max\left( \sqrt{\frac{\lambda_{r+1}}{N}}, \frac{2\|L_N - L_m\|_{HS}}{\Delta - \|L_N - L_m\|_{HS}} \right)$.*

**Remark** To utilize the result in Theorem 4, we consider the case when $\lambda_{r+1} = O(N/m)$ and $\Delta = \Omega(1)$. We have

$$\|L_N^{1/2}\Delta H L_N^{1/2}\|_2 \leq O\left( \max\left[ \frac{1}{\sqrt{m}}\|L_N - L_m\|_{HS}, \|L_N - L_m\|_{HS}^2 \right] \right).$$

Obviously, in order to achieve $O(1/m)$ bound for $\|L_N^{1/2}\Delta H L_N^{1/2}\|_2$, we need an $O(1/\sqrt{m})$ bound for $\|L_N - L_m\|_{HS}$, which is given by the following theorem.

**Theorem 5** *For $\kappa(\mathbf{x}, \mathbf{x}) \leq 1, \forall \mathbf{x} \in \mathcal{X}$, with a probability $1 - N^{-3}$, we have*

$$\|L_N - L_m\|_{HS} \leq \frac{2\ln(2N^3)}{m} + \sqrt{\frac{2\ln(2N^3)}{m}}.$$

Theorem 5 directly follows from Lemma 2 of [18]. Therefore, by assuming the conditions in Theorem 1 and combining results from Theorems 3, 4, and 5, we immediately have $\|K - \widehat{K}_r\|_2 \leq O(N/m)$. Combining this bound with the result in Theorem 2 and using the union bound, we have, with a probability $1 - 3N^{-3}$, $\Lambda(f_m^*) \leq 3\Lambda(f_N^*) + \frac{C}{\lambda}\left(\epsilon^2 + \frac{1}{m} + e^{-N}\right)$. We complete the proof of Theorem 1 by using the fact $e^{-N} < 1/N \leq 1/m$.

## 5  Empirical Studies

To verify our theoretical findings, we evaluate the empirical performance of the Nyström method and random Fourier features for large-scale kernel learning. Table 1 summarizes the statistics of the six data sets used in our study, including two for regression and four for classification. Note that datasets CPU, CENSUS, ADULT and FOREST were originally used in [13] to verify the effectiveness of random Fourier features. We evaluate the classification performance by accuracy, and the performance of regression by mean square error of the testing data.

We use uniform sampling in the Nyström method owing to its simplicity. We note that the empirical performance of the Nyström method may be improved by using a different implementation [21, 11]. We download the codes from the website `http://berkeley.intel-research.net/arahimi/c/random-features` for the implementation of random Fourier features. A RBF kernel is used for both methods and for all the datasets. A ridge regression package from [13] is used for the two regression tasks, and LIBSVM [5] is used for the classification tasks. All parameters are selected by a 5-fold cross validation. All experiments are repeated ten times, and prediction performance averaged over ten trials is reported.

Figure 2 shows the performance of both methods with varied number of random samples. Note that for large datasets (i.e., COVTYPE and FOREST), we restrict the maximum number of random samples to 200 because of the high computational cost. We observed that for all the data sets, the Nyström method outperforms random Fourier features [6]. Moreover, except for COVTYPE with 10 random samples, the Nyström method performs significantly better than random Fourier features, according to $t$-tests at 95% significance level. We finally evaluate that whether the large eigengap condition, the key assumption for our main theoretical result, holds for the data sets. Due to the large size, except for CPU, we compute the eigenvalues of kernel matrix based on $10,000$ randomly selected examples from each dataset. As shown in Figure 3 (eigenvalues are in logarithm scale), we observe that the eigenvalues drop very quickly as the rank increases, leading to a significant gap between the top eigenvalues and the remaining eigenvalues.

## 6  Conclusion and Discussion

We study two methods for large-scale kernel learning, i.e., the Nyström method and random Fourier features. One key difference between these two approaches is that the Nyström method uses data

Table 1: Statistics of data Sets

| TASK | DATA | # TRAIN | # TEST | #Attr. | TASK | DATA | # TRAIN | # TEST | #Attr. |
|------|------|---------|--------|--------|------|------|---------|--------|--------|
| Reg. | CPU | 6,554 | 819 | 21 | Class. | COD-RNA | 59,535 | 271,617 | 8 |
| Reg. | CENSUS | 18,186 | 2,273 | 119 | Class. | COVTYPE | 464,810 | 116,202 | 54 |
| Class. | ADULT | 32,561 | 16,281 | 123 | Class. | FOREST | 522,910 | 58,102 | 54 |

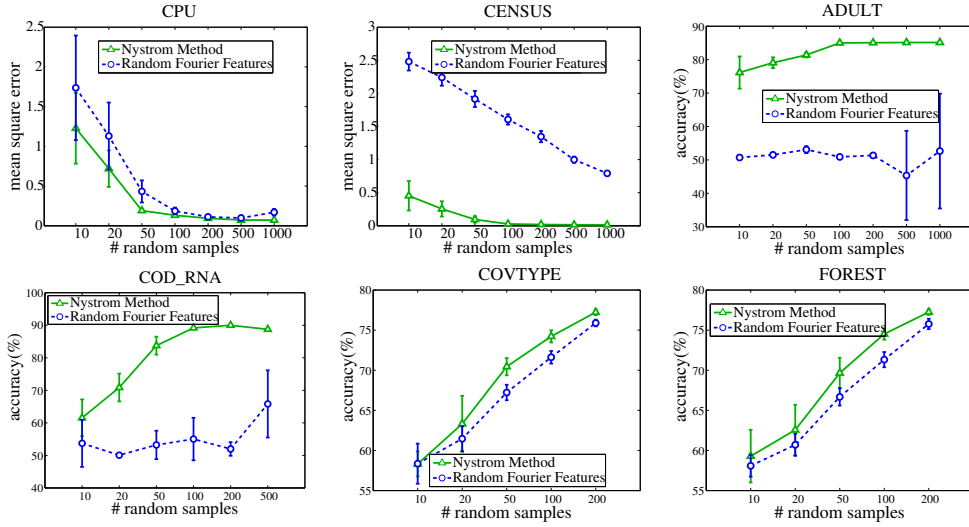

Figure 2: Comparison of the Nymström method and random Fourier features. For regression tasks, the mean square error (with std.) is reported, and for classification tasks, accuracy (with std.) is reported.

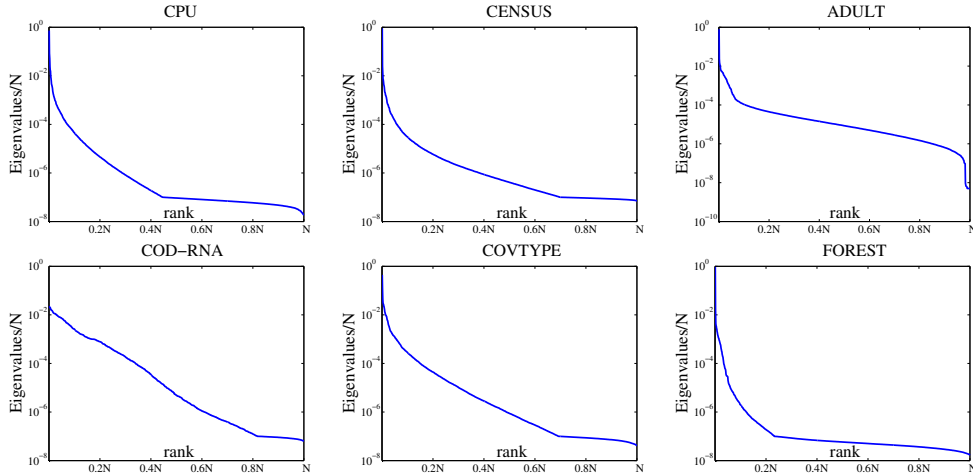

Figure 3: The eigenvalue distributions of kernel matrices. $N$ is the number of examples used to compute eigenvalues.

dependent basis functions while random Fourier features introduce data independent basis functions. This difference leads to an improved analysis for kernel learning approaches based on the Nyström method. We show that when there is a large eigengap of kernel matrix, the approximation error of Nyström method can be improved to $O(1/m)$, leading to a significantly better generalization performance than random Fourier features. We verify the claim by an empirical study.

As implied from our study, it is important to develop data dependent basis functions for large-scale kernel learning. One direction we plan to explore is to improve random Fourier features by making the sampling data dependent. This can be achieved by introducing a rejection procedure that rejects the sample Fourier components when they do not align well with the top eigenfunctions estimated from the sampled data.

## Acknowledgments

This work was partially supported by ONR Award N00014-09-1-0663, NSF IIS-0643494, NSFC (61073097) and 973 Program (2010CB327903).

## Footnotes

[1] We choose the bound based on spectral norm according to the discussion in [6].

[2] The improved bound obtained in the paper for the Nystrom method is valid for any kernel matrix that satisfies the eigengap condition.

[3]We use $\mathcal{H}_\mathcal{D}$, instead of $\mathcal{H}_\kappa$ in (3), owing to the representer theorem [16].

[4]Note that the scales of the two axes in Figure 1(a) are different.

[5]It is possible to achieve a better generalization error bound of $O(N^{-p/(p+1)})$ by assuming the eigenvalues of kernel matrix follow a $p$-power law [10]. However, large eigengap doest not immediately indicate power law distribution for eigenvalues and and consequently a better generalization error.

[6]We note that the classification performance of ADULT data set reported in Figure 2 does not match with the performance reported in [13]. Given the fact that we use the code provided by [13] and follow the same cross validation procedure, we believe our result is correct. We did not use the KDDCup dataset because of the problem of oversampling, as pointed out in [13].

# References

[1] A. Azran and Z. Ghahramani. Spectral methods for automatic multiscale data clustering. In *CVPR*, pages 190–197, 2006.

[2] F. R. Bach and M. I. Jordan. Learning spectral clustering. Technical Report UCB/CSD-03-1249, EECS Department, University of California, Berkeley, 2003.

[3] F. R. Bach and M. I. Jordan. Predictive low-rank decomposition for kernel methods. In *ICML*, pages 33–40, 2005.

[4] P. L. Bartlett, O. Bousquet, and S. Mendelson. Local rademacher complexities. *Annals of Statistics*, pages 44–58, 2002.

[5] C. Chang and C. Lin. Libsvm: a library for support vector machines. *TIST*, 2(3):27, 2011.

[6] C. Cortes, M. Mohri, and A. Talwalkar. On the impact of kernel approximation on learning accuracy. In *AISTAT*, pages 113–120, 2010.

[7] O. Dekel, S. Shalev-Shwartz, and Y. Singer. The forgetron: A kernel-based perceptron on a fixed budget. In *NIPS*, 2005.

[8] P. Drineas and M. W. Mahoney. On the nystrom method for approximating a gram matrix for improved kernel-based learning. *JMLR*, 6:2153–2175, 2005.

[9] J. Kivinen, A. J. Smola, and R. C. Williamson. Online learning with kernels. *IEEE Transactions on Signal Processing*, pages 2165–2176, 2004.

[10] V. Koltchinskii. *Oracle Inequalities in Empirical Risk Minimization and Sparse Recovery Problems*. Springer, 2011.

[11] S. Kumar, M. Mohri, and A. Talwalkar. Ensemble nystrom method. *NIPS*, pages 1060–1068, 2009.

[12] U. Luxburg. A tutorial on spectral clustering. *Statistics and Computing*, 17(4):395–416, 2007.

[13] A. Rahimi and B. Recht. Random features for large-scale kernel machines. *NIPS*, pages 1177–1184, 2007.

[14] A. Rahimi and B. Recht. Weighted sums of random kitchen sinks: Replacing minimization with randomization in learning. *NIPS*, pages 1313–1320, 2009.

[15] W. Rudin. *Fourier analysis on groups*. Wiley-Interscience, 1990.

[16] B. Schölkopf and A. J. Smola. *Learning with Kernels: Support Vector Machines, Regularization, Optimization, and Beyond*. MIT Press, 2001.

[17] T. Shi, M. Belkin, and B. Yu. Data spectroscopy: eigenspace of convolution operators and clustering. *The Annals of Statistics*, 37(6B):3960–3984, 2009.

[18] S. Smale and D.-X. Zhou. Geometry on probability spaces. *Constructive Approximation*, 30(3):311–323, 2009.

[19] G. W. Stewart and J. Sun. *Matrix Perturbation Theory*. Academic Press, 1990.

[20] C. Williams and M. Seeger. Using the nystrom method to speed up kernel machines. *NIPS*, pages 682–688, 2001.

[21] K. Zhang, I. W. Tsang, and J. T. Kwok. Improved nystrom low-rank approximation and error analysis. In *ICML*, pages 1232–1239, 2008.

